# Small-World Phenomena and the Dynamics of Information

**Jon Kleinberg**
Department of Computer Science
Cornell University
Ithaca NY 14853

## 1   Introduction

The problem of searching for information in networks like the World Wide Web can be approached in a variety of ways, ranging from centralized indexing schemes to decentralized mechanisms that navigate the underlying network without knowledge of its global structure. The decentralized approach appears in a variety of settings: in the behavior of users browsing the Web by following hyperlinks; in the design of *focused crawlers* [4, 5, 8] and other agents that explore the Web's links to gather information; and in the search protocols underlying decentralized peer-to-peer systems such as Gnutella [10], Freenet [7], and recent research prototypes [21, 22, 23], through which users can share resources without a central server.

In recent work, we have been investigating the problem of decentralized search in large information networks [14, 15]. Our initial motivation was an experiment that dealt directly with the search problem in a decidedly pre-Internet context: Stanley Milgram's famous study of the *small-world phenomenon* [16, 17]. Milgram was seeking to determine whether most pairs of people in society were linked by short chains of acquaintances, and for this purpose he recruited individuals to try forwarding a letter to a designated "target" through people they knew on a first-name basis. The starting individuals were given basic information about the target — his name, address, occupation, and a few other personal details — and had to choose a single acquaintance to send the letter to, with goal of reaching the target as quickly as possible; subsequent recipients followed the same procedure, and the chain closed in on its destination. Of the chains that completed, the median number of steps required was six — a result that has since entered popular culture as the "six degrees of separation" principle [11].

Milgram's experiment contains two striking discoveries — that short chains are pervasive, and that people are able to *find* them. This latter point is concerned precisely with a type of decentralized navigation in a *social network*, consisting of people as nodes and links joining pairs of people who know each other. From an algorithmic perspective, it is an interesting question to understand the structure of networks in which this phenomenon emerges — in which message-passing with purely local information can be efficient.

**Networks that Support Efficient Search.**   A model of a "navigable" network requires a few basic features. It should contain short paths among all (or most) pairs of nodes. To be non-trivial, its structure should be partially known and

partially unknown to its constituent nodes; in this way, information about the known parts can be used to construct paths that make use of the unknown parts as well. This is clearly what was taking place in Milgram's experiments: participants, using the information available to them, were estimating which of their acquaintances would lead to the shortest path through the full social network. Guided by these observations, we turned to the work of Watts and Strogatz [25], which proposes a model of "small-world networks" that very concisely incorporates these features. A simple variant of their basic model can be described as follows. One starts with a $p$-dimensional lattice, in which nodes are joined only to their nearest neighbors. One then adds $k$ directed *long-range* links out of each node $v$, for a constant $k$; the endpoint of each link is chosen uniformly at random. Results from the theory of random graphs can be used to show that with high probability, there will be short paths connecting all pairs of nodes (see e.g. [3]); at the same time, the network will locally retain a lattice-like structure. Asymptotically, our criterion for "shortness" of paths is what one obtains from this and similar random constructions: there should be paths whose lengths are bounded by a polynomial in $\log n$, where $n$ is the number of nodes. We will refer to such a function as *polylogarithmic*.

This network model, a superposition of a lattice and a set of long-range links, is a natural one in which to study the behavior of a decentralized search algorithm. The algorithm knows the structure of the lattice; it starts from a node $s$, and is told the coordinates of a target node $t$. It successively traverses links of the network so as to reach the target as quickly as possible; but, crucially, it does not know the long-range links out of any node that it has not yet visited. In addition to moving forward across directed links, the algorithm may travel in reverse across any link that it has already followed in the forward direction; this allows it to back up when it does not want to continue exploring from its current node. One can view this as hitting the "back button" on a Web browser — or returning the letter to its previous holder in Milgram's experiments, with instructions that he or she should try someone else. We say that the algorithm has *search time* $Y(n)$ if, on a randomly generated $n$-node network with $s$ and $t$ chosen uniformly at random, it reaches the target $t$ in at most $Y(n)$ steps with probability at least $1 - \varepsilon(n)$, for a function $\varepsilon(\cdot)$ going to 0 with $n$.

The first result in [15] is that no decentralized algorithm can achieve a polylogarithmic search time in this network model — even though, with high probability, there are paths of polylogarithmic length joining all pairs of nodes. However, if we generalize the model slightly, then it can support efficient search. Specifically, when we construct a long-range link $(v, w)$ out of $v$, we do not choose $w$ uniformly at random; rather, we choose it with probability proportional to $d^{-\alpha}$, where $d$ is the lattice distance from $v$ to $w$. In this way, the long-range links become correlated to the geometry of the lattice. We show in [15] that when $\alpha$ is equal to $p$, the dimension of the underlying lattice, then a decentralized greedy algorithm achieves search time proportional to $\log^2 n$; and for any other value of $\alpha$, there is no decentralized algorithm with a polylogarithmic search time.

Recent work by Zhang, Goel, and Govindan [26] has shown how the distribution of links associated with the optimal value of $\alpha$ may lead to improved performance for decentralized search in the Freenet peer-to-peer system. Adamic, Lukose, Puniyani, and Huberman [2] have recently considered a variation of the decentralized search problem in a network that has essentially no known underlying structure; however, when the number of links incident to nodes follows a power-law distribution, then a search strategy that seeks high-degree nodes can be effective. They have applied their results to the Gnutella system, which exhibits such a structure. In joint work with Kempe and Demers [12], we have studied how distributions that are

inverse-polynomial in the distance between nodes can be used in the design of *gossip protocols* for spreading information in a network of communicating agents.

The goal of the present paper is to consider more generally the problem of decentralized search in networks with partial information about the underlying structure. While a lattice makes for a natural network backbone, we would like to understand the extent to which the principles underlying efficient decentralized search can be abstracted away from a lattice-like structure. We begin by considering networks that are generated from a hierarchical structure, and show that qualitatively similar results can be obtained; we then discuss a general model of *group structures*, which can be viewed as a simultaneous generalization of lattices and hierarchies.

We refer to $k$, the number of out-links per node, as the *out-degree* of the model. The technical details of our results — both in the statements of the results and the proofs — are simpler when we allow the out-degree to be polylogarithmic, rather than constant. Thus we describe this case first, and then move on to the case in which each node has only a constant number of out-links.

## 2   Hierarchical Network Models

In a number of settings, nodes represent objects that can be classified according to a hierarchy or taxonomy; and nodes are more likely to form links if they belong to the same small sub-tree in the hierarchy, indicating they are more closely related.

To construct a network model from this idea, we represent the hierarchy using a complete $b$-ary tree $T$, where $b$ is a constant. Let $V$ denote the set of leaves of $T$; we use $n$ to denote the size of $V$, and for two leaves $v$ and $w$, we use $h(v, w)$ to denote the height of the least common ancestor of $v$ and $w$ in $T$. We are also given a monotone non-increasing function $f(\cdot)$ that will determine link probabilities. For each node $v \in V$, we create a random link to $w$ with probability proportional to $f(h(v, w))$; in other words, the probability of choosing $w$ is equal to $f(h(v, w))/\sum_{x \neq v} f(h(v, x))$. We create $k$ links out of each node $v$ this way, choosing the endpoint $w$ each time independently and with repetition allowed. This results in a graph $G$ on the set $V$.

For the analysis in this section, we will take the out-degree to be $k = c \log^2 n$, for a constant $c$. It is important to note that the tree $T$ is used only in the generation process of $G$; neither the edges nor the non-leaf nodes of $T$ appear in $G$. (By way of contrast, the lattice model in [15] included both the long-range links *and* the nearest-neighbor links of the lattice itself.) When we use the term "node" without any qualification, we are referring to nodes of $G$, and hence to leaves of $T$; we will use "internal node" to refer to non-leaf nodes of $T$.

We refer to the process producing $G$ as a *hierarchical model with exponent $\alpha$* if the function $f(h)$ grows asymptotically like $b^{-\alpha h}$:

$$\lim_{h \to \infty} \frac{f(h)}{b^{-\alpha' h}} = 0 \text{ for all } \alpha' < \alpha \text{ and } \lim_{h \to \infty} \frac{b^{-\alpha'' h}}{f(h)} = 0 \text{ for all } \alpha'' > \alpha.$$

There are several natural interpretations for a hierarchical network model. One is in terms of the World Wide Web, where $T$ is a *topic hierarchy* such as *www.yahoo.com*. Each leaf of $T$ corresponds to a Web page, and its path from the root specifies an increasingly fine-grained description of the page's topic. Thus, a particular leaf may be associated with *Science/Computer_Science/Algorithms* or with *Arts/Music/Opera*. The linkage probabilities then have a simple meaning — they are based on the distance between the topics of the pages, as measured by the height of their least common ancestor in the topic hierarchy. A page on *Sci-*

*ence/Computer_Science/Algorithms* may thus be more likely to link to one on *Science/Computer_Science/Machine_Learning* than to one on *Arts/Music/Opera*. Of course, the model is a strong simplification, since topic structures are not fully hierarchical, and certainly do not have uniform branching and depth. It is worth noting that a number of recent models for the link structure of the Web, as well as other relational structures, have looked at different ways in which similarities in content can affect the probability of linkage; see e.g. [1, 6, 9].

Another interpretation of the hierarchical model is in terms of Milgram's original experiment. Studies performed by Killworth and Bernard [13] showed that in choosing a recipient for the letter, participants were overwhelmingly guided by one of two criteria: similarity to the target in geography, or similarity to the target in occupation. If one views the lattice as forming a simple model for geographic factors, the hierarchical model can similarly be interpreted as forming a "topic hierarchy" of occupations, with individuals at the leaves. Thus, for example, the occupations of "banker" and "stock broker" may belong to the same small sub-tree; since the target in one of Milgram's experiments was a stock broker, it might therefore be a good strategy to send the letter to a banker. Independently of our work here, Watts, Dodds, and Newman have recently studied hierarchical structures for modeling Milgram's experiment in social networks [24].

We now consider the search problem in a graph $G$ generated from a hierarchical model: A decentralized algorithm has knowledge of the tree $T$, and knows the location of a target leaf that it must reach; however, it only learns the structure of $G$ as it visits nodes. The exponent $\alpha$ determines how the structures of $G$ and $T$ are related; how does this affect the navigability of $G$? In the analysis of the lattice model [15], the key property of the optimal exponent was that, from any point, there was a reasonable probability of a long-range link that halved the distance to the target. We make use of a similar idea here: when $\alpha = 1$, there is always a reasonable probability of finding a long-range link into a strictly smaller sub-tree containing the target. As mentioned above, we focus here on the case of polylogarithmic out-degree, with the case of constant out-degree deferred until later.

**Theorem 2.1** *(a) There is a hierarchical model with exponent $\alpha = 1$ and poly-logarithmic out-degree in which a decentralized algorithm can achieve search time $O(\log n)$.*

*(b) For every $\alpha \neq 1$, there is no hierarchical model with exponent $\alpha$ and polylogarithmic out-degree in which a decentralized algorithm can achieve polylogarithmic search time.*

Due to space limitations, we omit proofs from this version of the paper. Complete proofs may be found in the extended version, which is available on the author's Web page (http://www.cs.cornell.edu/home/kleinber/).

To prove (a), we show that when the search is at a node $v$ whose least common ancestor with the target has height $h$, there is a high probability that $v$ has a link into the sub-tree of height $h-1$ containing the target. In this way, the search reaches the target in logarithmically many steps. To prove (b), we exhibit a sub-tree $T'$ containing the target such that, with high probability, it takes any decentralized algorithm more than a polylogarithmic number of steps to find a link into $T'$.

## 3  Group Structures

The analysis of the search problem in a hierarchical model is similar to the analysis of the lattice-based approach in [15], although the two types of models seem

superficially quite different. It is natural to look for a model that would serve as a simultaneous generalization of each.

Consider a collection of individuals in a social network, and suppose that we know of certain *groups* to which individuals belong — people who live in the same town, or work in the same profession, or have some other affiliation in common. We could imagine that people are more likely to be connected if they both belong to the same small group. In a lattice-based model, there may be a group for each subset of lattice points contained in a common ball (grouping based on proximity); in a hierarchical model, there may be a group for each subset of leaves contained in a common sub-tree. We now discuss the notion of a *group structure*, to make this precise; we follow a model proposed in joint work with Kempe and Demers [12], where we were concerned with designing *gossip protocols* for lattices and hierarchies. A technically different model of *affiliation networks*, also motivated by these types of issues, has been studied recently by Newman, Watts, and Strogatz [18].

A *group structure* consists of an underlying set $V$ of *nodes*, and a collection of subsets of $V$ (the *groups*). The collection of groups must include $V$ itself; and it must satisfy the following two properties, for constants $\lambda < 1$ and $\beta > 1$.

(i) If $R$ is a group of size $q \geq 2$ containing a node $v$, then there is a group $R' \subseteq R$ containing $v$ that is strictly smaller than $R$, but has size at least $\lambda q$.

(ii) If $R_1, R_2, R_3, \ldots$ are groups that all have size at most $q$ and all contain a common node $v$, then their union has size at most $\beta q$.

The reader can verify that these two properties hold for the collection of balls in a lattice, as well as for the collection of sub-trees in a hierarchy. However, it is easy to construct examples of group structures that do not arise in this way from lattices or hierarchies.

Given a group structure $(V, \{R_i\})$, and a monotone non-increasing function $f(\cdot)$, we now consider the following process for generating a graph on $V$. For two nodes $v$ and $w$, we use $q(v, w)$ to denote the minimum size of a group containing both $v$ and $w$. (Note that such a group must exist, since $V$ itself is a group.) For each node $v \in V$, we create a random link to $w$ with probability proportional to $f(q(v, w))$; repeating this $k$ times independently yields $k$ links out of $v$. We refer to this as a *group-induced model with exponent* $\alpha$ if $f(q)$ grows asymptotically like $q^{-\alpha}$:

$$\lim_{h \to \infty} \frac{f(q)}{q^{-\alpha'}} = 0 \text{ for all } \alpha' < \alpha \text{ and } \lim_{h \to \infty} \frac{q^{-\alpha''}}{f(q)} = 0 \text{ for all } \alpha'' > \alpha.$$

A decentralized search algorithm in such a network is given knowledge of the full group structure, and must follow links of $G$ to a designated target $t$. We now state an analogue of Theorem 2.1 for group structures.

**Theorem 3.1** *(a) For every group structure, there is a group-induced model with exponent $\alpha = 1$ and polylogarithmic out-degree in which a decentralized algorithm can achieve search time $O(\log n)$.*

*(b) For every $\alpha < 1$, there is no group-induced model with exponent $\alpha$ and polylogarithmic out-degree in which a decentralized algorithm can achieve polylogarithmic search time.*

Notice that in a hierarchical model, the smallest group (sub-tree) containing two nodes $v$ and $w$ has size $b^{h(v,w)}$, and so Theorem 3.1(a) implies Theorem 2.1(a). Similarly, on a lattice, the smallest group (ball) containing two nodes $v$ and $w$ at

lattice distance $d$ has size $\Theta(d^p)$, and so Theorem 3.1(a) implies a version of the result from [15], that efficient search is possible in a lattice model when nodes form links with probability $d^{-p}$. (In the version of the lattice result implied here, there are no nearest-neighbor links at all; but each node has a polylogarithmic number of out-links.)

The proof of Theorem 3.1(a) closely follows the proof of Theorem 2.1(a). We consider a node $v$ — the current point in the search — for which the smallest group containing $v$ and the target $t$ has size $q$. Using group structure properties (i) and (ii), we show there is a high probability that $v$ has a link into a group containing $t$ of size between $\lambda^2 q$ and $\lambda q$. In this way, the search passes through groups containing $t$ of sizes that diminish geometrically, and hence it terminates in logarithmic time.

Note that Theorem 3.1(b) only considers exponents $\alpha < 1$. This is because there exist group-induced models with exponents $\alpha > 1$ in which decentralized algorithms can achieve polylogarithmic search time. For example, consider an undirected graph $G^*$ in which each node has 3 neighbors, and each pair of nodes can be connected by a path of length $O(\log n)$. It is possible to define a group structure satisfying properties (i) and (ii) in which each edge of $G^*$ appears as a 2-node group; but then, a graph $G$ generated from a group-induced model with a very large exponent $\alpha$ will contain all edges of $G^*$ with high probability, and a decentralized search algorithm will be able to follow these edges directly to construct a short path to the target.

However, a lower bound for the case $\alpha > 1$ can be obtained if we place one additional restriction on the group structure. Give a group structure $(V, \{R_i\})$, and a cut-off value $q$, we define a graph $H(q)$ on $V$ by joining any two nodes that belong to a common group of size at most $q$. Note that $H(q)$ is not a random graph; it is defined simply in terms of the group structure and $q$. We now argue that if many pairs of nodes are far apart in $H(q)$, for a suitably large value of $q$, then a decentralized algorithm cannot be efficient when $\alpha > 1$.

**Theorem 3.2** *Let $(V, \{R_i\})$ be a group structure. Suppose there exist constants $\gamma, \delta > 0$ so that a constant fraction of all pairs of nodes have shortest-path distance $\Omega(n^\delta)$ in $H(n^\gamma)$. Then for every $\alpha > 1$, there is no group-induced model on $(V, \{R_i\})$ with exponent $\alpha$ and a polylogarithmic number of out-links per node in which a decentralized algorithm can achieve polylogarithmic search time.*

Notice this property holds for group structures arising from both lattices and hierarchies; in a lattice, a constant fraction of all pairs in $H(n^{1/2p})$ have distance $\Omega(n^{1/2p})$, while in a hierarchy, the graph $H(n^\gamma)$ is disconnected for every $\gamma < 1$.

# 4 Nodes with a Constant Number of Out-Links

Thus far, by giving each node more than a constant number of out-links, we have been able to design very simple search algorithms in networks generated according to the optimal exponent $\alpha$. From each node, there is a way to make progress toward the target node $t$, and so the structure of the graph $G$ funnels the search towards its destination. When the out-degree is constant, however, things get much more complicated. First of all, with high probability, many nodes will have all their links leading "away" from the target in the hierarchy. Second, there is a constant probability that the target $t$ will have *no in-coming links*, and so the whole task of finding $t$ becomes ill-defined. This indicates that the statement of the results themselves in this case will have to be somewhat different.

In this section, we work with a hierarchical model, and construct graphs with con-

stant out-degree $k$; the value of $k$ will need to be sufficiently large in terms of other parameters of the model. It is straightforward to formulate an analogue of our results for group structures, but we do not go into the details of this here.

To deal with the problem that $t$ itself may have no incoming links, we relax the search problem to that of finding a *cluster* of nodes containing $t$. In a topic-based model of Web pages, for example, we can consider $t$ as a representative of a desired type of page, with goal being to find any page of this type. Thus, we are given a complete $b$-ary tree $T$, where $b$ is a constant; we let $L$ denote the set of leaves of $T$, and $m$ denote the size of $L$. We place $r$ nodes at each leaf of $T$, forming a set $V$ of $n = mr$ nodes total. We then define a graph $G$ on $V$ as in Section 2: for a non-increasing function $f(\cdot)$, we create $k$ links out of each node $v \in V$, choosing $w$ as an endpoint with probability proportional to $f(h(v,w))$. As before, we refer to this process as a hierarchical model with exponent $\alpha$, for the appropriate value of $\alpha$. We refer to each set of $r$ nodes at a common leaf of $T$ as a *cluster*, and define the *resolution* of the hierarchical model to be the value $r$.

A decentralized algorithm is given knowledge of $T$, and a target node $t$; it must reach any node in the cluster containing $t$. Unlike the previous algorithms we have developed, which only moved forward across links, the algorithm we design here will need to make use of the ability to travel in reverse across any link that it has already followed in the forward direction. Note also that we cannot easily reduce the current search problem to that of Section 2 by collapsing clusters into "super-nodes," since there are not necessarily links joining nodes within the same cluster.

The search task clearly becomes easier as the resolution of the model (i.e. the size of clusters) becomes larger. Thus, our goal is to achieve polylogarithmic search time in a hierarchical model with polylogarithmic resolution.

**Theorem 4.1** *(a) There is a hierarchical model with exponent $\alpha = 1$, constant out-degree, and polylogarithmic resolution in which a decentralized algorithm can achieve polylogarithmic search time.*

*(b) For every $\alpha \neq 1$, there is no hierarchical model with exponent $\alpha$, constant out-degree, and polylogarithmic resolution in which a decentralized algorithm can achieve polylogarithmic search time.*

The search algorithm used to establish part (a) operates in phases. It begins each phase $j$ with a collection of $\Theta(\log n)$ nodes all belonging to the sub-tree $T_j$ that contains the target $t$ and whose root is at depth $j$. During phase $j$, it explores outward from each of these nodes until it has discovered a larger but still polylogarithmic-sized set of nodes belonging to $T_j$. From among these, there is a high probability that at least $\Theta(\log n)$ have links into the smaller sub-tree $T_{j+1}$ that contains $t$ and whose root is at depth $j + 1$. At this point, phase $j + 1$ begins, and the process continues until the cluster containing $t$ is found.

### Acknowledgments

My thinking about models for Web graphs and social networks has benefited greatly from discussions and collaboration with Dimitris Achlioptas, Avrim Blum, Duncan Callaway, Michelle Girvan, John Hopcroft, David Kempe, Ravi Kumar, Tom Leighton, Mark Newman, Prabhakar Raghavan, Sridhar Rajagopalan, Steve Strogatz, Andrew Tomkins, Eli Upfal, and Duncan Watts. The research described here was supported in part by a David and Lucile Packard Foundation Fellowship, an ONR Young Investigator Award, NSF ITR/IM Grant IIS-0081334, and NSF Faculty Early Career Development Award CCR-9701399.

# References

[1] D. Achlioptas, A. Fiat, A. Karlin, F. McSherry, "Web search via hub synthesis," *Proc. 42nd IEEE Symp. on Foundations of Computer Science*, 2001.

[2] L. Adamic, R. Lukose, A. Puniyani, B. Huberman, "Search in Power-Law Networks," *Phys. Rev. E*, 64 46135 (2001)

[3] B. Bollobás, F. Chung, "The diameter of a cycle plus a random matching," *SIAM J. Disc. Math.* 1(1988).

[4] S. Chakrabarti, M. van den Berg, B. Dom, "Focused crawling: A new approach to topic-specific Web resource discovery," *Proc. 8th Intl. World Wide Web Conf.*, 1999.

[5] J. Cho, H. Garcia-Molina, L. Page, "Efficient Crawling Through URL Ordering," *Proc. 7th Intl. World Wide Web Conf.*, 1998.

[6] D. Cohn and T. Hofmann, "The Missing Link – A Probabilistic Model of Document Content and Hypertext Connectivity," *Adv. Neural Inf. Proc. Sys. (NIPS)* 13, 2000.

[7] I. Clarke, O. Sandberg, B. Wiley, T. Hong, "Freenet: A Distributed Anonymous Information Storage and Retrieval System," *International Workshop on Design Issues in Anonymity and Unobservability*, 2000.

[8] M. Diligenti, F.M. Coetzee, S. Lawrence, C.L. Giles, M. Gori, "Focused Crawling Using Context Graphs," *Proc. 26th Intl. Conf. on Very Large Databases (VLDB)*, 2000.

[9] L. Getoor, N. Friedman, D. Koller, and B. Taskar. "Learning Probabilistic Models of Relational Structure," *Proc. 18th International Conference on Machine Learning*, 2001.

[10] Gnutella. http://gnutella.wego.com.

[11] J. Guare, *Six Degrees of Separation: A Play* (Vintage Books, New York, 1990).

[12] D. Kempe, J. Kleinberg, A. Demers. "Spatial gossip and resource location protocols," *Proc. 33rd ACM Symp. on Theory of Computing*, 2001.

[13] P. Killworth, H. Bernard, "Reverse small world experiment," *Social Networks* 1(1978).

[14] J. Kleinberg. "Navigation in a Small World." *Nature* 406(2000).

[15] J. Kleinberg. "The small-world phenomenon: An algorithmic perspective." *Proc. 32nd ACM Symposium on Theory of Computing*, 2000. Also appears as Cornell Computer Science Technical Report 99-1776 (October 1999).

[16] M. Kochen, Ed., *The Small World* (Ablex, Norwood, 1989).

[17] S. Milgram, "The small world problem," *Psychology Today* 1(1967).

[18] M. Newman, D. Watts, S. Strogatz, "Random graph models of social networks," *Proc. Natl. Acad. Sci.*, to appear.

[19] A. Oram, editor, *Peer-to-Peer: Harnessing the Power of Disruptive Technologies* O'Reilly and Associates, 2001.

[20] A. Puniyani, R. Lukose, B. Huberman, "Intentional Walks on Scale Free Small Worlds," HP Labs Information Dynamics Group, at http://www.hpl.hp.com/shl/.

[21] S. Ratnasamy, P. Francis, M. Handley, R. Karp, S. Shenker, "A Scalable Content-Addressable Network," *Proc. ACM SIGCOMM*, 2001

[22] A. Rowstron, P. Druschel, "Pastry: Scalable, distributed object location and routing for large-scale peer-to-peer systems," *Proc. 18th IFIP/ACM International Conference on Distributed Systems Platforms (Middleware 2001)*, 2001.

[23] I. Stoica, R. Morris, D. Karger, F. Kaashoek, H. Balakrishnan, "Chord: A Scalable Peer-to-peer Lookup Service for Internet Applications," *Proc. ACM SIGCOMM*, 2001

[24] D. Watts, P. Dodds, M. Newman, personal communication, December 2001.

[25] D. Watts, S. Strogatz, "Collective dynamics of small-world networks," *Nature* 393(1998).

[26] H. Zhang, A. Goel, R. Govindan, "Using the Small-World Model to Improve Freenet Performance," *Proc. IEEE Infocom*, 2002.
